# Aggregating Classification Accuracy across Time: Application to Single Trial EEG

**Steven Lemm** *
Intelligent Data Analysis Group,
Fraunhofer Institute FIRST,
Kekulestr. 7
12489 Berlin,
Germany

**Christin Schäfer**
Intelligent Data Analysis Group,
Fraunhofer Institute FIRST,
Kekulestr. 7
12489 Berlin,
Germany

**Gabriel Curio**
Neurophysics Group, Dept. of Neurology,
Campus Benjamin Franklin,
Charité,University Medicine Berlin,
Hindenburgdamm 20,
12200 Berlin,
Germany

## Abstract

We present a method for binary on-line classification of triggered but temporally blurred events that are embedded in noisy time series in the context of on-line discrimination between left and right imaginary hand-movement. In particular the goal of the binary classification problem is to obtain the decision, as fast and as reliably as possible from the recorded EEG single trials. To provide a probabilistic decision at every time-point $t$ the presented method gathers information from two distinct sequences of features across time. In order to incorporate decisions from prior time-points we suggest an appropriate weighting scheme, that emphasizes time instances, providing a higher discriminatory power between the instantaneous class distributions of each feature, where the discriminatory power is quantified in terms of the Bayes error of misclassification.

The effectiveness of this procedure is verified by its successful application in the 3rd BCI competition. Disclosure of the data after the competition revealed this approach to be superior with single trial error rates as low as 10.7, 11.5 and 16.7% for the three different subjects under study.

## 1 Introduction

The ultimate goal of brain-computer interfacing (BCI) is to translate human intentions into a control signal for a device, such as a computer application, a wheelchair or a neuroprosthesis (e.g. [20]). Most pursued approaches utilize the accompanying EEG-rhythm perturbation in order to distinguish between single trials (STs) of left and right hand imaginary movements e.g. [8, 11, 14, 21]. Up to now there are just a few published approaches utilizing additional features, such as slow cortical potential, e.g. [3, 4, 9]

This paper describes the algorithm that has been successfully applied in the 2005 international data analysis competition on BCI-tasks [2] (data set IIIb) for the on-line discrimina-

tion between imagined left and right hand movement. The objective of the competition was to detect the respective motor intention as early and as reliably as possible. Consequently, the competing algorithms have to solve the on-line discrimination task as based on information on the event onset. Thus it is not within the scope of the competition to solve the problem of detecting the event onset itself.

We approach this problem by applying an algorithm that combines the different characteristics of two features: the modulations of the ongoing rhythmic activity and the slow cortical Movement Related Potential (MRP). Both features are differently pronounced over time and exhibit a large trial to trial variability and can therefore be considered as temporally blurred. Consequently, the proposed method combines at one hand the MRP with the oscillatory feature and on the other hand gather information across time as introduced in [8,16]. More precisely, at each time point we estimate probabilistic models on the labeled training data - one for each class and feature - yielding a sequence of weak instantaneous classifiers, i.e. posterior class distributions. The classification of an unlabeled ST is then derived by weighted combination of these weak probabilistic classifiers using linear combination according to their instantaneous discriminatory power.

The paper is organized as follows: section II describes the feature and its extraction, In section III introduces the probabilistic model as well as the combinatorial framework to gather information from the different features across time. In section III the results on the competition data are given, followed by a brief conclusion.

## 2 Feature

### 2.1 Neurophysiology

The human perirolandic sensorimotor cortices show rhythmic macroscopic EEG oscillations ($\mu$-rhythm) [6], with spectral peak energies around $10\,\mathrm{Hz}$ (localized predominantly over the postcentral somatosensory cortex) and $20\,\mathrm{Hz}$ (over the precentral motor cortex). Modulations of the $\mu$-rhythm have been reported for different physiological manipulations, e.g., by motor activity, both actual and imagined [7,13,18], as well as by somatosensory stimulation [12]. Standard trial averages of $\mu$-rhythm power show a sequence of attenuation, termed event-related desynchronization (ERD) [13], followed by a rebound (event-related synchronization: ERS) which often overshoots the pre-event baseline level [15].

In case of sensorimotor cortical processes accompanying finger movements Babiloni et al. [1] demonstrated that movement related potentials (MRPs) and ERD indeed show up with different spatio-temporal activation patterns across primary (sensori-)motor cortex (M1), supplementary motor area (SMA) and the posterior parietal cortex (PP). Most importantly, the ERD response magnitude did not correlate with the amplitude of the negative MRPs slope.

In the subsequent we will combine both features. Thus, in order to extract the rhythmic information we map the EEG to the time-frequency domain by means of Morlet wavelets [19], whereas the slow cortical MRP are extracted by the application of a low pass filter, in form of a simple moving average filter.

### 2.2 Extraction

Let $X = [x[1], \ldots, x[T]]$ denote the EEG signal of one single trial (ST) of length $T$, recorded from the two bipolar channels C3 and C4, i.e. $x[t] = [\mathrm{C3}[t], \mathrm{C4}[t]]^{\mathrm{T}}$. The label information about the corresponding motor intention of a ST is denoted by $Y \in \{L, R\}$. For information obtain from observations until time $s \leq T$, we will make use of subscript $_{|s}$ throughout this paper, e.g. $X_{|s}$ refers to $[x[1], \ldots, x[s]]$. This observational horizon becomes important with respect to the causality of the feature extraction process, especially in order to ensure the causality of filter operations we have to restrict the algorithm to a certain observational horizon. Note that $X_{|T}$ denotes a completely observed ST. However for notational convenience we will omit the index $_{|T}$ in case of complete observations.

Considering ERD as a feature for ST classifications we model the hand-specific time course of absolute $\mu$-rhythm amplitudes over both sensorimotor cortices. Therefore we utilize the time-frequency representations of the ST at two different frequency bands $(\alpha, \beta)$, obtained by convolution of the EEG signal with complex Morlet wavelets [19]. Using the notation $\Psi_\alpha$, and $\Psi_\beta$ for a wavelet centered at the individual spectral peak in the alpha (8-12Hz) and the beta (16-24Hz) frequency domain, the ERD feature of a ST, observed until time $s$ is calculated as:

$$
\begin{aligned}
\text{ERD}_{|s} &= \left[ \text{erd}_{|s}[1], \ldots, \text{erd}_{|s}[s] \right], \\
&\text{with} \\
\text{erd}_{|s}[t] &= \left[ \begin{array}{c} |(\text{C3}_{|s} * \Psi_\alpha)[t]| \\ |(\text{C4}_{|s} * \Psi_\alpha)[t]| \\ |(\text{C3}_{|s} * \Psi_\beta)[t]| \\ |(\text{C4}_{|s} * \Psi_\beta)[t]| \end{array} \right].
\end{aligned}
\tag{1}
$$

In a similar manner we define the ST feature for the MRP by convolution with a moving average filter of length 11, abbreviated as MA(11).

$$
\begin{aligned}
\text{MRP}_{|s} &= \left[ \text{mrp}_{|s}[1], \ldots, \text{mrp}_{|s}[s] \right], \\
&\text{with} \\
\text{mrp}_{|s}[t] &= \left[ \begin{array}{c} (\text{C3}_{|s} * \text{MA}(11))[t] \\ (\text{C4}_{|s} * \text{MA}(11))[t] \end{array} \right].
\end{aligned}
\tag{2}
$$

According to (1) and (2) the $k$-th labeled, observed STs for training, i.e. $\left(X^{(k)}, Y^{(k)}\right)$ maps to a STs in feature space, namely $(\text{MRP}^{(k)}, \text{ERD}^{(k)})$.

## 3 Probabilistic Classification Model

Before we start with the model description, we briefly introduce two concepts from Bayesian decision theory. Therefore let $p(x|\mu_y, \Sigma_y), y \in \{L, R\}$ denote the PDFs of two multivariate Gaussian distributions with different means and covariance matrices $(\mu_y, \Sigma_y)$ for two classes, denoted by $L$ and $R$. Given the two class-conditional distribution models, and under the assumption of a class prior of $P(y) = \frac{1}{2}$, $y \in \{L, R\}$, and given an observation $x$, the posterior class distribution according to Bayes formula is given by

$$
p(y|x, \mu_L, \Sigma_L, \mu_R, \Sigma_R) = \frac{p(x|\mu_y, \Sigma_y)}{p(x|\mu_L, \Sigma_L) + p(x|\mu_R, \Sigma_R)}.
\tag{3}
$$

Furthermore the discriminative power between these two distributions can be estimated using the Bayes error of misclassification [5]. In case of distinct class covariance matrices, the Bayes error cannot be calculated directly. However, by using the Chernoff bound [5] we can derive an upper bound and finally approximate the discriminative power $w$ between the two distributions by

$$
2w \cong 1 - \min_{0 \leq \gamma \leq 1} \int p(x|\mu_L, \Sigma_L)^\gamma p(x|\mu_R, \Sigma_R)^{1-\gamma} dx.
\tag{4}
$$

In case of Gaussian distributions the above integral can be expressed in a closed form [5], such that the minimum solution can be easily obtained (see also [16]).

Based on these two necessary concepts, we will now introduce our probabilistic classification method. Therefore we first model the class-conditional distribution of each feature at each time instance as multivariate Gaussian distribution. Hence at each time instance we estimate the class means and the class covariance matrices in the feature space, based on the mapped training STs, i.e. $\text{ERD}^{(k)}, \text{MRP}^{(k)}$. Thus from $\text{erd}^{(k)}[t]$ we obtain the following two class-conditional sets of parameters:

$$
\begin{aligned}
\mu_y[t] &= \text{E}\left[ \text{erd}^{(k)}[t] \right]_{Y^{(k)}=y} \tag{5} \\
\Sigma_y[t] &= \text{Cov}\left[ \text{erd}^{(k)}[t] \right]_{Y^{(k)}=y}, \ y \in \{L, R\}. \tag{6}
\end{aligned}
$$

For convenience we summarize the estimated model parameters for the ERD feature as $\Theta[t] := (\mu_L[t], \Sigma_L[t], \mu_R[t], \Sigma_R[t])$, whereas $\Xi[t] := (\eta_L[t], \Gamma_L[t]), \eta_R[t], \Gamma_R[t])$ denote the class means and the covariance matrices obtained in the similar manner from $\mathrm{mrp}^{(k)}[t]$. Given an arbitrary observation $x$ from the appropriate domain, applying Bayes formula as introduced in (3), yields a posterior distribution for each feature:

$$p\left(y\middle|\mathrm{erd}, \Theta[t]\right), \quad \mathrm{erd} \in \mathbb{R}^4 \tag{7}$$

$$p\left(y\middle|\mathrm{mrp}, \Xi[t]\right), \quad \mathrm{mrp} \in \mathbb{R}^2. \tag{8}$$

Additionally, according to (4) we get approximations of the discriminative power $w[t]$ and $v[t]$ of the ERP resp. MRP feature at every time instance.

In order to finally derive the classification of an unlabeled single trial at a certain time $s \leq T$, we incorporate knowledge from all preceding samples $t \leq s$, i.e. we make the classification based on the causally extracted features: $\mathrm{ERD}_{|s}$ and $\mathrm{MRP}_{|s}$. Therefore we first apply (7) and (8) given the observations $\mathrm{erd}_{|s}[t]$ resp. $\mathrm{mrp}_{|s}[t]$ in order to obtain the class posteriors based on observations until $s \leq T$. Secondly we combine these class posteriors with one another across time by taking the expectation under the distributions $w$ and $v$, i.e.

$$c(y, s) = \sum_{t \leq s} \frac{w[t] \cdot p\left(y\middle|\mathrm{erd}_{|s}[t], \Theta[t]\right) + v[t] \cdot p\left(y\middle|\mathrm{mrp}_{|s}[t], \Xi[t]\right)}{\sum_{t \leq s} w[t] + v[t]}. \tag{9}$$

As described in [16] this yields an evidence accumulation over time about the decision process. Strictly speaking Eq. (9) gives the expectation value that the ST, observed until time $s$, is generated by either one of the class models (L or R), until time $s$. Due to the submission requirements of the competition the final decision at time $s$ is

$$C[s] = 1 - 2 \cdot c(\mathrm{L}, s), \tag{10}$$

where a positive or negative sign refers to right or left movement, while the magnitude indicates the confidence in the decision on a scale between 0 and 1.

## 4 Application

### 4.1 Competition data

The EEG from two bipolar channels (C3, C4) was provided with bandfilter settings of 0.5 to 30 Hz and sampled at 128 Hz. The data consist of recordings from three different healthy subjects. Except for the first data set, each contains 540 labeled (for training) and 540 unlabeled trials (for competition) of imaginary hand movements, with an equal number of left and right hand trials (first data set provides just 320 trials each). Each trial has a duration of 7 s: after a 3 s preparation period a visual cue is presented for one second, indicating the demanded motor intention. This is followed by another 3 s for performing the imagination task (for details see [2]). The particular competition data was provided by the Dept. of Med. Informatics, Inst. for Biomed. Eng., Univ. of Techn. Graz. The specific competition task is to provide an on-line discrimination between left and right movements for the unlabeled STs for each subject based on the information obtained from the labeled trials. More precisely, at every time instance in the interval from 3 to 7 seconds a strictly causal decision about the intended motor action and its confidence must be supplied. After competition deadline, based on the disclosure of the labels $Y^{(k)}$ for the previously unlabeled STs the output $C^{(k)}[t]$ of the methods were evaluated using the time course of the mutual information (MI) [17], i.e.

$$\mathrm{MI}[t] = \frac{1}{2} \log_2 \left(\mathrm{SNR}[t] + 1\right) \tag{11}$$

$$\mathrm{SNR}[t] = \frac{\left(\mathrm{E}\left[C^{(k)}[t]\right]_{Y^{(k)}=L} - \mathrm{E}\left[C^{(k)}[t]\right]_{Y^{(k)}=R}\right)^2}{2\left(\mathrm{Var}\left[C^{(k)}[t]\right]_{Y^{(k)}=L} + \mathrm{Var}\left[C^{(k)}[t]\right]_{Y^{(k)}=R}\right)} \tag{12}$$

More precisely, since the general objective of the competition was to obtain the single trial classification as fast and as accurate as possible, the maximum steepness of the MI was

considered as final evaluation criterion, i.e.

$$\max_{t\geq 3.5} \frac{\text{MI}[t]}{t-3\text{s}}. \tag{13}$$

Note, that the feature extraction relies on a few hyperparameters, i.e. the center frequency and the width of the wavelets, as well as the length of the MA filter. All those parameters were obtained by model selection using a leave-one-out cross-validation scheme of the classification performance on the training data.

## 4.2   Results and Discussion

As proposed in section 3 we estimated the class-conditional Gaussian distributions cf. (5) – (8). The resulting posterior distributions were then combined according to (9) in order to obtain the final classification of the unlabeled STs. After disclosure of the label information our method turned out to succeed with a MI steepness (cf. (13)) of $0.17, 0.44$ and $0.35$ for the individual subjects. Table 4.2 summarizes the results in terms of the achieved minimum binary classification error, the maximum MI, and the maximum steepness of MI for each subject and each competitor in the competition.

|     | min. error rate[%] | | | max. MI [bit] | | | max. MI/t [bit/s] | | |
| --- | --- | --- | --- | --- | --- | --- | --- | --- | --- |
|     | O3 | S4 | X11 | O3 | S4 | X11 | O3 | S4 | X11 |
| 1. | **10.69** | **11.48** | 16.67 | **0.6027** | **0.6079** | **0.4861** | 0.1698 | **0.4382** | **0.3489** |
| 2. | 14.47 | 22.96 | 22.22 | 0.4470 | 0.2316 | 0.3074 | 0.1626 | 0.4174 | 0.1719 |
| 3. | 13.21 | 17.59 | **16.48** | 0.5509 | 0.3752 | 0.4675 | **0.2030** | 0.0936 | 0.1173 |
| 4. | 23.90 | 24.44 | 24.07 | 0.2177 | 0.2387 | 0.2173 | 0.1153 | 0.1218 | 0.1181 |
| 5. | 11.95 | 21.48 | 18.70 | 0.4319 | 0.3497 | 0.3854 | 0.1039 | 0.1490 | 0.0948 |
| 6. | **10.69** | 13.52 | 25.19 | 0.5975 | 0.5668 | 0.2437 | 0.1184 | 0.1516 | 0.0612 |
| 7. | 34.28 | 38.52 | 28.70 | 0.0431 | 0.0464 | 0.1571 | 0.0704 | 0.0229 | 0.0489 |

Table 1: Overall ranked results of the competing algorithms (first row corresponds to the proposed method) on the competition test data. For three different subjects (O3, S4 and X11) the table states different performance measures of classification accuracy (min. Error rate, max MI, steepness of MI), where the steepness of the MI was used as the objective in the competition. For a description of the 2.–7. algorithm please refer to [2].

The resulting time courses for the MI and the steepness of the MI are presented in the left panel of Fig. 1. For subject two and three, during the first 3.5 seconds (0.5 seconds after cue presentation) the classification is rather by chance, after 3.5 seconds a steep ascent in the classification accuracy can be observed, reflected by the raising MI. The maximum steepness for these two subjects is obtained quite early, between $3.6 - 3.8$s. In opposite, for subject one the maximum is achieved at 4.9 seconds, yielding a low steepness value. However, a low value is also found for the submission of all other competitors. Nevertheless, the MI constantly increases up to 0.64 Bit per trial at 7 seconds, which might indicate a delayed performance of subject one.

The right panel in Fig. 1 provides the weights $w[t]$ and $v[t]$, reflecting the Bayes error of misclassification cf. (4), that were used for the temporal integration process. For subject two one can clearly observe a switch in the regime between the ERP and the MRP feature at 5 seconds, as indicated by a crossing of the two weighting functions. From this we conclude that the steep increase in MI for this subject between 3 and 5 seconds is mainly due to the MRP feature, whereas the further improvement in the MI relies primarily on the ERD feature. Subject one provides nearly no discriminative MRP and the classification is almost exclusively based on the ERD feature. For subject three the constant low weights at all time instances, reveal the weak discriminative power of the estimated class-conditional distributions. However in Fig. 1 the advantage of the integration process across time can clearly be observed, as the MI is continuously increasing and the steepness of the MI is surprisingly high even for this subject.

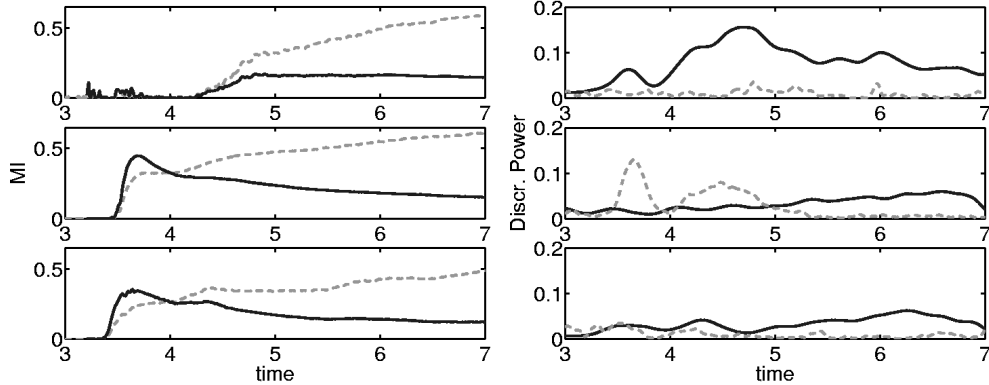

Figure 1: Left panel: time courses of the mutual information (light, dashed) and the competition criterion - steepness of mutual information (thin solid) cf. (13)- for the classification of the unlabeled STs is presented. Right panel: the time course of the weights reflecting the discriminative power (cf. (4)) at every time instance for the two different features (ERD - dark, solid; MRP - light dashed). In each panel the subjects O3, S4, X11 are arranged top down.

A comprehensive comparison of all submitted techniques to solve the specific task for data set IIIb of the BCI-competition is provided in [2] or available on the web [1]. Basically this evaluation reveals that the proposed algorithm outperforms all competing approaches.

# 5 Conclusion

We proposed a general Bayesian framework for temporal combination of sets of simple classifiers based on different features, which is applicable to any kind of sequential data providing a binary classification problems. Moreover, any arbitrary number of features can be combined in the proposed way of temporal weighting, by utilizing the estimated discriminative power over time.

Furthermore the estimation of the Bayes error of misclassification is not strictly linked to the chosen parametric form of the class-conditional distributions. For arbitrary distributions the Bayes error can be obtained for instance by statistical resampling approaches, such as Monte Carlo methods.

However for the successful application in the BCI-competition 2005 we chose Gaussian distribution for the sake of simplicity concerning two issues: estimating their parameters and obtaining their Bayes error. Note that although the combination of the classifiers across time is linear, the final classification model is non-linear, as the individual classifiers at each time instance are non-linear.For a discussion about linear vs. non-linear methods in the context of BCI see [10]. More precisely due to the distinct covariance matrices of the Gaussian distributions the individual decision boundaries are of quadratic form. In particular to solve the competition task we combined classifiers based on the temporal evolution of different neuro-physiological features, i.e. ERD and MRP. The resulting on-line classification model finally turned out to succeed for the single trial on-line classification of imagined hand movement in the BCI competition 2005.

**Acknowledgement:** This work was supported in part by the Bundesministerium für Bildung und Forschung (BMBF) under grant FKZ 01GQ0415 and by the DFG under grant SFB 618-B4. S. Lemm thanks Stefan Harmeling for valuable discussions.

## Footnotes

*steven.lemm@first.fhg.de

[1] ida.first.fhg.de/projects/bci/competition_iii/

# References

[1] C. Babiloni, F. Carducci, F. Cincotti, P. M. Rossini, C. Neuper, Gert Pfurtscheller, and F. Babiloni. Human movement-related potentials vs desynchronization of EEG alpha rhythm: A high-resolution EEG study. *NeuroImage*, 10:658–665, 1999.

[2] Benjamin Blankertz, Klaus-Robert Müller, Dean Krusienski, Gerwin Schalk, Jonathan R. Wolpaw, Alois Schlögl, Gert Pfurtscheller, José del R. Millán, Michael Schröder, and Niels Birbaumer. The BCI competition III: Validating alternative approachs to actual BCI problems. *IEEE Trans. Neural Sys. Rehab. Eng.*, 14(2):153–159, 2006.

[3] Guido Dornhege, Benjamin Blankertz, Gabriel Curio, and Klaus-Robert Müller. Combining features for BCI. In S. Becker, S. Thrun, and K. Obermayer, editors, *Advances in Neural Inf. Proc. Systems (NIPS 02)*, volume 15, pages 1115–1122, 2003.

[4] Guido Dornhege, Benjamin Blankertz, Gabriel Curio, and Klaus-Robert Müller. Increase information transfer rates in BCI by CSP extension to multi-class. In Sebastian Thrun, Lawrence Saul, and Bernhard Schölkopf, editors, *Advances in Neural Information Processing Systems*, volume 16, pages 733–740. MIT Press, Cambridge, MA, 2004.

[5] R.O. Duda, P.E. Hart, and D.G. Stork. *Pattern Classification*. John Wiley & Sons, New York, 2nd edition, 2001.

[6] R. Hari and R. Salmelin. Human cortical oscillations: a neuromagnetic view through the skull. *Trends in Neuroscience*, 20:44–9, 1997.

[7] H. Jasper and W. Penfield. Electrocorticograms in man: Effect of voluntary movement upon the electrical activity of the precentral gyrus. *Arch. Psychiatrie Zeitschrift Neurol.*, 183:163–74, 1949.

[8] Steven Lemm, Christin Schäfer, and Gabriel Curio. Probabilistic modeling of sensorimotor $\mu$ rhythms for classification of imaginary hand movements. *IEEE Trans. Biomed. Eng.*, 51(6):1077–1080, 2004.

[9] B.D. Mensh, J. Werfer, and H.S. Seung. Combining gamma-band power with slow cortical potentials to improve single-trial classification of electroencephalographic signals. *IEEE Trans. Biomed. Eng.*, 51(6):1052–6, 2004.

[10] Klaus-Robert Müller, Charles W. Anderson, and Gary E. Birch. Linear and non-linear methods for brain-computer interfaces. *IEEE Trans. Neural Sys. Rehab. Eng.*, 11(2):165–169, 2003.

[11] C. Neuper, A. Schlögl, and G. Pfurtscheller. Enhancement of left-right sensorimotor EEG differences during feedback-regulated motor imagery. *Journal Clin. Neurophysiol.*, 16:373–82, 1999.

[12] V. Nikouline, K. Linkenkaer-Hansen, Wikström; H., M. Kesäniemi, E. Antonova, R. Ilmoniemi, and J. Huttunen. Dynamics of mu-rhythm suppression caused by median nerve stimulation: a magnetoencephalographic study in human subjects. *Neurosci. Lett.*, 294, 2000.

[13] G. Pfurtscheller and A. Arabibar. Evaluation of event-related desynchronization preceding and following voluntary self-paced movement. *Electroencephalogr. Clin. Neurophysiol.*, 46:138–46, 1979.

[14] G. Pfurtscheller, C. Neuper, D. Flotzinger, and M. Pregenzer. EEG-based discrimination between imagination of right and left hand movement. *Electroenceph. clin. Neurophysiol.*, 103:642–51, 1997.

[15] S. Salenius, A. Schnitzler, R. Salmelin, V. Jousmäki, and R. Hari. Modulation of human cortical rolandic rhythms during natural sensorimotor tasks. *NeuroImage*, 5:221–8, 1997.

[16] Christin Schäfer, Steven Lemm, and Gabriel Curio. Binary on-line classification based on temporally integrated information. In Claus Weihs and Wolfgang Gaul, editors, *Proceedings of the 28th annual conference of the Gesellschaft für Klassifikation*, pages 216–223, 2005.

[17] A. Schlögl, R. Scherer C. Keinrath, and G. Pfurtscheller. Information transfer of an EEG-based brain-computer interface. In *Proc. First Int. IEEE EMBS Conference on Neural Engineering*, pages 641–644, 2003.

[18] A. Schnitzler, S. Salenius, R. Salmelin, V. Jousmäki, and R. Hari. Involvement of primary motor cortex in motor imagery: a neuromagnetic study. *NeuroImage*, 6:201–8, 1997.

[19] C. Torrence and G.P. Compo. A practical guide to wavelet analysis. *Bull. Am. Meterol.*, 79:61–78, 1998.

[20] Jonathan R. Wolpaw, Niels Birbaumer, Dennis J. McFarland, Gert Pfurtscheller, and Theresa M. Vaughan. Brain-computer interfaces for communication and control. *Clin. Neurophysiol.*, 113:767–791, 2002.

[21] J.R. Wolpaw and D.J. McFarland. Multichannel EEG-based brain-computer communication. *Electroenceph. clin. Neurophysiol.*, 90:444–9, 1994.
